# Optimizing spatio-temporal filters for improving Brain-Computer Interfacing

**Guido Dornhege[1], Benjamin Blankertz[1], Matthias Krauledat[1,3],**
**Florian Losch[2], Gabriel Curio[2] and Klaus-Robert Müller[1,3]**

[1]Fraunhofer FIRST.IDA, Kekuléstr. 7, 12 489 Berlin, Germany

[2]Campus Benjamin Franklin, Charité University Medicine Berlin,
Hindenburgdamm 30, 12 203 Berlin, Germany.

[3]University of Potsdam, August-Bebel-Str. 89, 14 482 Germany

*{dornhege,blanker,kraulem,klaus}@first.fhg.de,*
*{florian-philip.losch,gabriel.curio}@charite.de*

## Abstract

Brain-Computer Interface (BCI) systems create a novel communication channel from the brain to an output device by bypassing conventional motor output pathways of nerves and muscles. Therefore they could provide a new communication and control option for paralyzed patients. Modern BCI technology is essentially based on techniques for the classification of single-trial brain signals. Here we present a novel technique that allows the simultaneous optimization of a spatial and a spectral filter enhancing discriminability of multi-channel EEG single-trials. The evaluation of 60 experiments involving 22 different subjects demonstrates the superiority of the proposed algorithm. Apart from the enhanced classification, the spatial and/or the spectral filter that are determined by the algorithm can also be used for further analysis of the data, e.g., for source localization of the respective brain rhythms.

## 1 Introduction

Brain-Computer Interface (BCI) research aims at the development of a system that allows direct control of, e.g., a computer application or a neuroprosthesis, solely by human intentions as reflected in suitable brain signals, cf. [1, 2, 3, 4, 5, 6, 7, 8, 9]. We will be focussing on noninvasive, electroencephalogram (EEG) based BCI systems. Such devices can be used as tools of communication for the disabled or for healthy subjects that might be interested in exploring a new path of man-machine interfacing, say when playing BCI operated computer games.

The classical approach to establish EEG-based control is to set up a system that is controlled by a specific EEG feature which is known to be susceptible to conditioning and to let the subjects learn the voluntary control of that feature. In contrast, the Berlin Brain-Computer Interface (BBCI) uses well established motor competences in control paradigms and a machine learning approach to extract subject-specific discriminability patterns from high-dimensional features. This approach has the advantage that the long subject training needed in the operant conditioning approach is replaced by a short calibration measurement

(20 minutes) and machine training (1 minute). The machine adapts to the specific characteristics of the brain signals of each subject, accounting for the high inter-subject variability. With respect to the topographic patterns of brain rhythm modulations the Common Spatial Patterns (CSP) (see [10]) algorithm has proven to be very useful to extract subject-specific, discriminative spatial filters. On the other hand the frequency band on which the CSP algorithm operates is either selected manually or unspecifically set to a broad band filter, cf. [10, 5]. Obviously, a simultaneous optimization of a frequency filter with the spatial filter is highly desirable. Recently, in [11] the CSSP algorithm was presented, in which very simple frequency filters (with one delay tap) for each channel are optimized together with the spatial filters. Although the results showed an improvement of the CSSP algorithm over CSP, the flexibility of the frequency filters is very limited. Here we present a method that allows to simultaneously optimize an arbitrary FIR filter within the CSP analysis. The proposed algorithm outperforms CSP and CSSP on average, and in cases where a separation of the discriminative rhythm from dominating non-discriminative rhythms is of importance, a considerable increase of classification accuracy can be achieved.

## 2 Experimental Setup

In this paper we investigate data from 60 EEG experiments with 22 different subjects. All experiments included so called training sessions which are used to train subject-specific classifiers. Many experiments also included feedback sessions in which the subject could steer a cursor or play a computer game like *brain-pong* by BCI control. Data from feedback sessions are not used in this a-posteriori study since they depend on an intricate interaction of the subject with the original classification algorithm.

In the experimental sessions used for the present study, labeled trials of brain signals were recorded in the following way: The subjects were sitting in a comfortable chair with arms lying relaxed on the armrests. All 4.5–6 seconds one of 3 different visual stimuli indicated for 3–3.5 seconds which mental task the subject should accomplish during that period. The investigated mental tasks were imagined movements of the left hand ($l$), the right hand ($r$), and one foot ($f$). Brain activity was recorded from the scalp with multi-channel EEG amplifiers using 32, 64 resp. 128 channels. Besides EEG channels, we recorded the electromyogram (EMG) from both forearms and the leg as well as horizontal and vertical electrooculogram (EOG) from the eyes. The EMG and EOG channels were used exclusively to make sure that the subjects performed no real limb or eye movements correlated with the mental tasks that could directly (artifacts) or indirectly (afferent signals from muscles and joint receptors) be reflected in the EEG channels and thus be detected by the classifier, which operates on the EEG signals only. Between 120 and 200 trials for each class were recorded. In this study we investigate only binary classifications, but the results can be expected to safely transfer to the multi-class case.

## 3 Neurophysiological Background

According to the well established model called homunculus, first described by [12], for each part of the human body there exists a corresponding region in the motor and somatosensory area of the neocortex. The 'mapping' from the body to the respective brain areas preserves in big parts topography, i.e., neighboring parts of the body are almost represented in neighboring parts of the cortex. While the region of the feet is located at the center of the vertex, the left hand is represented lateralized on the right hemisphere and the right hand on the left hemisphere. Brain activity during rest and wakefulness is describable by different rhythms located over different brain areas. These rhythms reflect functional states of different neuronal cortical networks and can be used for brain-computer interfacing. These rhythms are blocked by movements, independent of their active, passive or reflexive origin. Blocking effects are visible bilaterally but pronounced contralaterally in the cortical area that corresponds to the moved limb. This attenuation of brain rhythms is

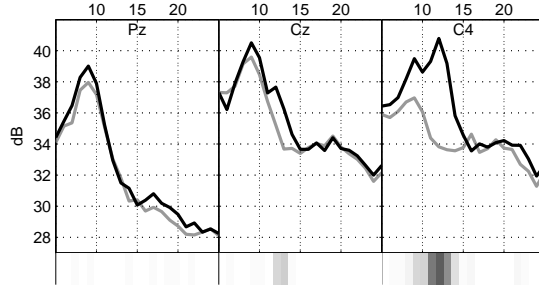

Figure 1: The plot shows the spectra for one subject during left hand (light line) and foot (dark line) motor imagery between 5 and 25 Hz at scalp positions Pz, Cz and C4. In both central channels two peaks, one at 8 Hz and one at 12 Hz are visible. Below each channel the $r^2$-value which measures discriminability is added. It indicates that the second peak contains more discriminative information.

termed event-related desynchronization (ERD), see [13]. Over sensorimotor cortex a so called idle- or $\mu$-rhythm can be measured in the scalp EEG. The most common frequency band of $\mu$-rhythm is about 10 Hz (precentral $\alpha$- or $\mu$-rhythm, [14]). Jasper and Penfield ([12]) described a strictly local so called beta-rhythm about 20 Hz over human motor cortex in electrocorticographic recordings. In Scalp EEG recording one can find $\mu$-rhythm over motor areas mixed and superimposed by 20 Hz-activity. In this context $\mu$-rhythm is sometimes interpreted as a subharmonic of cortical faster activity. These brain rhythms described above are of cortical origin but the role of a thalomo-cortical pacemaker has been discussed since the first description of EEG by Berger ([15]) and is still a point of discussion. Lopes da Silva ([16]) showed that cortico-cortical coherence is much larger than thalamo-cortical. However, since the focal ERD in the motor and/or sensory cortex can be observed even when a subject is only imagining a movement or sensation in the specific limb, this feature can well be used for BCI control. The discrimination of the imagination of movements of left hand vs. right hand vs. foot is based on the topography of the attenuation of the $\mu$ and/or $\beta$ rhythm.

There are two problems when using ERD features for BCI control:

(1) The strength of the sensorimotor idle rhythms as measured by scalp EEG is known to vary strongly between subjects. This introduces a high intersubject variability on the accuracy with which an ERD-based BCI system works. There is another feature independent from the ERD reflecting imagined or intended movements, the movement related potentials (MRP), denoting a negative DC shift of the EEG signals in the respective cortical regions. See [17, 18] for an investigation of how this feature can be exploited for BCI use and combined with the ERD feature. This combination strategy was able to greatly enhance classification performance in offline studies. In this paper we focus only on improving the ERD-based classification, but all the improvements presented here can also be used in the combined algorithm.

(2) The precentral $\mu$-rhythm is often superimposed by the much stronger posterior $\alpha$-rhythm, which is the idle rhythm of the visual system. It is best articulated with eyes closed, but also present in awake and attentive subjects, see Fig. 1 at channel Pz. Due to volume conduction the posterior $\alpha$-rhythm interferes with the precentral $\mu$-rhythm in the EEG channels over motor cortex. Hence a $\mu$-power based classifier is susceptible to modulations of the posterior $\alpha$-rhythm that occur due to fatigue, change in attentional focus while performing tasks, or changing demands of visual processing. When the two rhythms have different spectral peaks as in Fig. 1, channels Cz and C4, a suitable frequency filter can help to weaken the interference. The optimization of such a filter integrated in the CSP algorithm is addressed in this paper.

## 4  Spatial Filter - the CSP Algorithm

The common spatial pattern (CSP) algorithm ([19]) is very useful in calculating spatial filters for detecting ERD effects ([20]) and for ERD-based BCIs, see [10], and has been extended to multi-class problems in [21]. Given two distributions in a high-dimensional space, the (supervised) CSP algorithm finds directions (i.e., spatial filters) that maximize

variance for one class and at the same time minimize variance for the other class. After having band-pass filtered the EEG signals to the rhythms of interest, high variance reflects a strong rhythm and low variance a weak (or attenuated) rhythm. Let us take the example of discriminating left hand vs. right hand imagery. According to Sec. 3, the spatial filter that focusses on the area of the left hand is characterized by a strong motor rhythm during imagination of right hand movements (left hand is in idle state), and by an attenuated motor rhythm during left hand imagination.

This criterion is exactly what the CSP algorithm optimizes: maximizing variance for the class of right hand trials and at the same time minimizing variance for left hand trials. Furthermore the CSP algorithm calculates the dual filter that will focus on the area of the right hand (and it will even calculate several filters for both optimizations by considering orthogonal subspaces).

The CSP algorithm is trained on labeled data, i.e., we have a set of trials $s_i$, $i = 1, 2, ...$, where each trial consists of several channels (as rows) and time points (as columns). A spatial filter $w \in I\!\!R^{\#channels}$ projects these trials to the signal $\hat{s}_i(w) = w^\top s_i$ with only one channel. The idea of CSP is to find a spatial filter $w$ such that the projected signal has high variance for one class and low variance for the other. In other words we maximize the variance for one class whereas the sum of the variances of both classes remains constant, which is expressed by the following optimization problem:

$$\max_w \quad \sum_{i:\text{Trial in Class 1}} var(\hat{s}_i(w)), \quad \text{s.t.} \quad \sum_i var(\hat{s}_i(w)) = 1, \tag{1}$$

where $var(\cdot)$ is the variance of the vector. An analoguous formulation can be formed for the second class.

Using the definition of the variance we simplify the problem to

$$\max_w \quad w^\top \Sigma_1 w, \quad s.t. \quad w^\top (\Sigma_1 + \Sigma_2) w = 1, \tag{2}$$

where $\Sigma_y$ is the covariance matrix of the trial-concatenated matrix of dimension [channels $\times$ concatenated time-points] belonging to the respective class $y \in \{1, 2\}$.

Formulating the dual problem we can find that the problem can be solved by calculating a matrix $Q$ and diagonal matrix $D$ with elements in $[0, 1]$ such that

$$Q\Sigma_1 Q^\top = D \quad \text{and} \quad Q\Sigma_2 Q^\top = I - D \tag{3}$$

and by choosing the highest and lowest eigenvalue.

Equation (3) can be accomplished in the following way. First we *whiten* the matrix $\Sigma_1 + \Sigma_2$, i.e., determine a matrix $P$ such that $P(\Sigma_1 + \Sigma_2)P^\top = I$ which is possible due to positive definiteness of $\Sigma_1 + \Sigma_2$. Then define $\hat{\Sigma}_y = P\Sigma_y P^\top$ and calculate an orthogonal matrix $R$ and a diagonal maxtrix $D$ by spectral theory such that $\hat{\Sigma}_1 = RDR^\top$. Therefore $\hat{\Sigma}_2 = R(I - D)R^\top$ since $\hat{\Sigma}_1 + \hat{\Sigma}_2 = I$ and $Q := R^\top P$ satisfies (3). The projection that is given by the $j$-th row of matrix $R$ has a relative variance of $d_j$ ($j$-th element of $D$) for trials of class 1 and relative variance $1 - d_j$ for trials of class 2. If $d_j$ is near 1 the filter given by the $j$-th row of $R$ maximizes variance for class 1, and since $1 - d_j$ is near 0, minimizes variance for class 2. Typically one would retain some projections corresponding to the highest eigenvalues $d_j$, i.e., CSPs for class 1, and some corresponding to the lowest eigenvalues, i.e., CSPs for class 2.

## 5 Spectral Filter

As discussed in Sec. 3 the content of discriminative information in different frequency bands is highly subject-dependent. For example the subject whose spectra are visualized in Fig. 1 shows a highly discriminative peak at 12 Hz whereas the peak at 8 Hz does not show good discrimination. Since the lower frequency peak is stronger a better performance in

classification can be expected, if we reduce the influence of the lower frequency peak for this subject. However, for other subjects the situation looks differently, i.e., the classification might fail if we exclude this information. Thus it is desirable to optimize a spectral filter for better discriminability. Here are two approaches to this task.

**CSSP.** In [11] the following was suggested: Given $s_i$ the signal $s_i^\tau$ is defined to be the signal $s_i$ delayed by $\tau$ timepoints. In CSSP the usual CSP approach is applied to the concatenation of $s_i$ and $s_i^\tau$ in the channel dimension, i.e., the delayed signals are treated as new channels. By this concatenation step the ability to neglect or emphasize specific frequency bands can be achieved and strongly depends on the choice of $\tau$ which can be accomplished by some validation approach on the training set. More complex frequency filters can be found by concatenating more delayed EEG-signals with several delays. In [11] it was concluded that in typical BCI situations where only small training sets are available, the choice of only one delay tap is most effective. The increased flexibility of a frequency filter with more delay taps does not trade off the increased complexity of the optimization problem.

**CSSSP.** The idea of our new CSSSP algorithm is to learn a complete global spatial-temporal filter in the spirit of CSP and CSSP.

A digital frequency filter consists of two sequences $a$ and $b$ with length $n_a$ and $n_b$ such that the signal $x$ is filtered to $y$ by

$$
\begin{aligned}
a(1)y(t) = \quad & b(1)x(t) + b(2)x(t-1) + ... + b(n_b)x(t-n_b-1) \\
- \quad & a(2)y(t-1) - ... - a(n_a)y(t-n_a-1)
\end{aligned}
$$

Here we restrict ourselves to FIR (finite impulse response) filters by defining $n_a = 1$ and $a = 1$. Furthermore we define $b(1) = 1$ and fix the length of $b$ to some $T$ with $T > 1$. By this restriction we resign some flexibility of the frequency filter but it allows us to find a suitable solution in the following way: We are looking for a real-valued sequence $b_{1,...,T}$ with $b(1) = 1$ such that the trials

$$
s_{i,b} = s_i + \sum_{\tau=2,...,T} b_\tau s_i^\tau \tag{4}
$$

can be classified better in some way. Using equation (1) we have to solve the problem

$$
\max_{w,b,b(1)=1} \quad \sum_{i:\text{Trial in Class 1}} var(\hat{s}_{i,b}(w)), \quad \text{s.t.} \quad \sum_i var(\hat{s}_{i,b}(w)) = 1, \tag{5}
$$

which can be simplified to

$$
\begin{aligned}
\max_{b,b(1)=1} \max_w \quad & w^\top \left( \sum_{\tau=0,...,T-1} \left( \sum_{j=1,...,T-\tau} b(j)b(j+\tau) \right) \Sigma_1^\tau \right) w, \\
\text{s.t.} \quad & w^\top \left( \sum_{\tau=0,...,T-1} \left( \sum_{j=1,...,T-\tau} b(j)b(j+\tau) \right) (\Sigma_1^\tau + \Sigma_2^\tau) \right) w = 1.
\end{aligned} \tag{6}
$$

where $\Sigma_y^\tau = E(\langle s_i(s_i^\tau)^\top + s_i^\tau s_i^\top | i : \text{Trial in Class } y \rangle)$, namely the correlation between the signal and the by $\tau$ timepoints delayed signal.

Since we can calculate for each $b$ the optimal $w$ by the usual CSP techniques (see equation (2) and (3)) a $(T-1)$-dimensional (b(1)=1) problem remains which we can solve with usual line-search optimization techniques if $T$ is not too large.

Consequently we get for each class a frequency band filter and a pattern (or similar to CSP more than one pattern by choosing the next eigenvectors).

However, with increasing $T$ the complexity of the frequency filter has to be controlled in order to avoid overfitting. This control is achieved by introducing a regularization term in

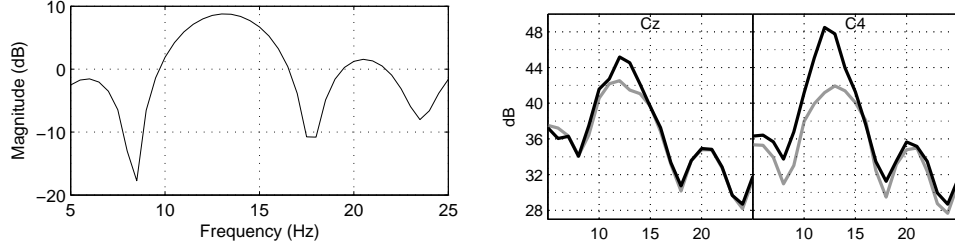

Figure 2: The plot on the left shows one learned frequency filter for the subject whose spectra was shown Fig. 1. In the plot on the right the resulting spectra are visualized after applying the frequency filter on the left. By this technique the classification error could be reduced from 12.9 % to 4.3 %.

the following way:

$$\max_{b, b(1)=1} \max_{w} \quad w^\top \left( \sum_{\tau=0,\dots,T-1} \left( \sum_{j=1,\dots,T-\tau} b(j)b(j+\tau) \right) \Sigma_1^\tau \right) w - C/T \|b\|_1,$$

$$s.t. \quad w^\top \left( \sum_{\tau=0,\dots,T-1} \left( \sum_{j=1,\dots,T-\tau} b(j)b(j+\tau) \right) (\Sigma_1^\tau + \Sigma_2^\tau) \right) w = 1. \tag{7}$$

Here $C$ is a non-negative regularization constant, which has to be chosen, e.g., by cross-validation. Since a sparse solution for $b$ is desired, we use the 1-norm in this formulation. With higher $C$ we get sparser solutions for $b$ until at one point the usual CSP approach remains, i.e., $b(1) = 1, b(m) = 0$ for $m > 1$. We call this approach *Common Sparse Spectral Spatial Pattern* (CSSSP) algorithm.

# 6 Feature Extraction, Classification and Validation

## 6.1 Feature Extraction

After choosing all channels except the EOG and EMG and a few of the outermost channels of the cap we apply a causal band-pass filter from 7–30 Hz to the data, which encompasses both the $\mu$- and the $\beta$-rhythm. For classification we extract the interval 500–3500 ms after the presented visual stimulus. To these trials we apply the original CSP ([10]) algorithm (see Sec. 4), the extended CSSP ([11]), and the proposed CSSSP algorithm (see Sec. 5). For CSSP we choose the best $\tau$ by leave-one-out cross validation on the training set. For CSSSP we present the results for different regularization constants $C$ with fixed $T = 16$. Here we use 3 patterns per class which leads to a 6-dimensional output signal. As a measure of the amplitude in the specified frequency band we calculate the logarithm of the variances of the spatio-temporally filtered output signals as feature vectors.

## 6.2 Classification and Validation

The presented preprocessing reduces the dimensionality of the feature vectors to six. Since we have 120 up to 200 samples per class for each data set, there is no need for regularization when using linear classifiers. When testing non-linear classification methods on these features, we could not observe any statistically significant gain for the given experimental setup when compared to Linear Discriminant Analysis (LDA) (see also [22, 6, 23]). Therefore we choose LDA for classification.

For validation purposes the (chronologically) first half of the data are used as training and the second half as test data.

# 7 Results

Fig. 2 shows one chosen frequency filter for the subject whose spectra are shown in Fig. 1 and the remaining spectrum after using this filter. As expected the filter detects that there

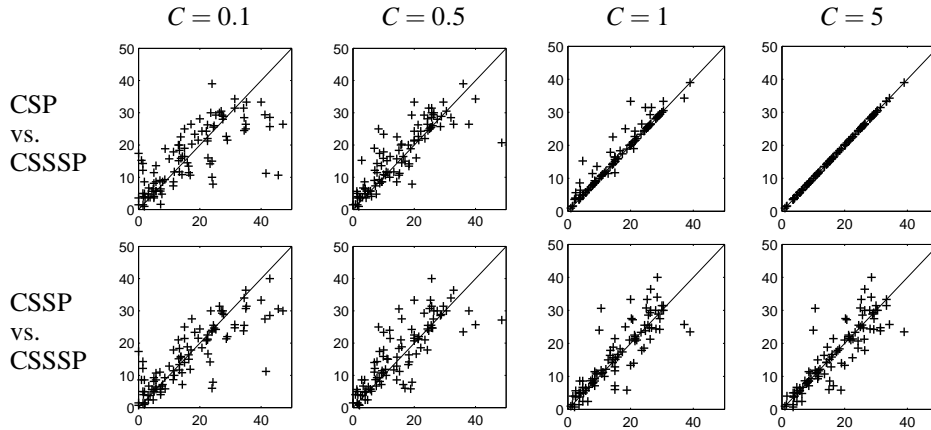

Figure 3: Each plots shows validation error of one algorithm against another, in row 1 that is CSP (*y*-axis) vs. CSSSP (*x*-axis), in row 2 that is CSSP (*y*-axis) vs. CSSSP (*x*-axis). In columns the regularization parameter of CSSSP is varied between 0.1, 0.5, 1 and 5. In each plot a cross above the diagonal marks a dataset where CSSSP outperforms the other algorithm.

is a high discrimination in frequencies at 12 Hz, but only a low discrimination in the frequency band at 8 Hz. Since the lower frequency peak is very predominant for this subject without having a high discrimination power, a filter is learned which drastically decreases the amplitude in this band, whereas full power at 12 Hz is retained.

Applied to all datasets and all pairwise class combinations of the datasets we get the results shown in Fig. 3. Only the results of those datasets are displayed whose classification accuracy exceeds 70 % for at least one classifier. First of all, it is obvious that a small choice of the regularization constant is problematic, since the algorithm tends to overfit. For high values CSSSP tends towards the CSP performance since using frequency filters is punished too hard. In between there is a range where CSSSP is better than CSP. Furthermore there are some datasets where the gain by CSSSP is huge.

Compared to CSSP the situation is similar, namely that CSSSP outperforms the CSSP in many cases and on average, but there are also a few cases, where CSSP is better.

An open issue is the choice of the parameter *C*. If we choose it constant at 1 for all datasets the figure shows that CSSSP will typically outperform CSP. Compared to CSSP both cases appear, namely that CSSP is better than CSSSP and vice versa.

A more refined way is to choose *C* individually for each dataset. One way to accomplish this choice is to perform cross-validations for a set of possible values of *C* and to select the *C* with minimum cross-validation error. We have done this, for example, for the dataset whose spectra are shown in Fig. 1. Here on the training set for *C* the value 0.3 is chosen. The classification error of CSSSP with this *C* is 4.3 %, whereas CSP has 12.9 % and CSSP 8.6 % classification error.

## 8    Concluding discussion

In past BCI research the CSP algorithm has proven to be very sucessful in determining spatial filters which extract discriminative brain rhythms. However the performance can suffer when a non-discriminative brain rhythm with an overlapping frequency range interferes. The presented CSSSP algorithm successful solves such problematic situations by optimizing simultaneously with the spatial filters a spectral filter. The trade-off between flexibility of the estimated frequency filter and the danger of overfitting is accounted for by a sparsity constraint which is weighted by a regularization constant. The successfulness of the proposed algorithm when compared to the original CSP and to the CSSP algorithm was demonstrated on a corpus of 60 EEG data sets recorded from 22 different subjects.

**Acknowledgments**  We thank S. Lemm for helpful discussions.  The studies were supported by BMBF-grants FKZ 01IBB02A and FKZ 01IBB02B, by the *Deutsche Forschungsgemeinschaft* (DFG), FOR 375/B1 and by the PASCAL Network of Excellence (EU # 506778).

## References

[1] J. R. Wolpaw, N. Birbaumer, D. J. McFarland, G. Pfurtscheller, and T. M. Vaughan, "Brain-computer interfaces for communication and control", *Clin. Neurophysiol.*, 113: 767–791, 2002.

[2] E. A. Curran and M. J. Stokes, "Learning to control brain activity: A review of the production and control of EEG components for driving brain-computer interface (BCI) systems", *Brain Cogn.*, 51: 326–336, 2003.

[3] A. Kübler, B. Kotchoubey, J. Kaiser, J. Wolpaw, and N. Birbaumer, "Brain-Computer Communication: Unlocking the Locked In", *Psychol. Bull.*, 127(3): 358–375, 2001.

[4] N. Birbaumer, N. Ghanayim, T. Hinterberger, I. Iversen, B. Kotchoubey, A. Kübler, J. Perelmouter, E. Taub, and H. Flor, "A spelling device for the paralysed", *Nature*, 398: 297–298, 1999.

[5] G. Pfurtscheller, C. Neuper, C. Guger, W. Harkam, R. Ramoser, A. Schlögl, B. Obermaier, and M. Pregenzer, "Current Trends in Graz Brain-computer Interface (BCI)", *IEEE Trans. Rehab. Eng.*, 8(2): 216–219, 2000.

[6] B. Blankertz, G. Curio, and K.-R. Müller, "Classifying Single Trial EEG: Towards Brain Computer Interfacing", in: T. G. Diettrich, S. Becker, and Z. Ghahramani, eds., *Advances in Neural Inf. Proc. Systems (NIPS 01)*, vol. 14, 157–164, 2002.

[7] L. Trejo, K. Wheeler, C. Jorgensen, R. Rosipal, S. Clanton, B. Matthews, A. Hibbs, R. Matthews, and M. Krupka, "Multimodal Neuroelectric Interface Development", *IEEE Trans. Neural Sys. Rehab. Eng.*, (11): 199–204, 2003.

[8] L. Parra, C. Alvino, A. C. Tang, B. A. Pearlmutter, N. Yeung, A. Osman, and P. Sajda, "Linear spatial integration for single trial detection in encephalography", *NeuroImage*, 7(1): 223–230, 2002.

[9] W. D. Penny, S. J. Roberts, E. A. Curran, and M. J. Stokes, "EEG-Based Communication: A Pattern Recognition Approach", *IEEE Trans. Rehab. Eng.*, 8(2): 214–215, 2000.

[10] H. Ramoser, J. Müller-Gerking, and G. Pfurtscheller, "Optimal spatial filtering of single trial EEG during imagined hand movement", *IEEE Trans. Rehab. Eng.*, 8(4): 441–446, 2000.

[11] S. Lemm, B. Blankertz, G. Curio, and K.-R. Müller, "Spatio-Spectral Filters for Improved Classification of Single Trial EEG", *IEEE Trans. Biomed. Eng.*, 52(9): 1541–1548, 2005.

[12] H. Jasper and W. Penfield, "Electrocorticograms in man: Effects of voluntary movement upon the electrical activity of the precentral gyrus", *Arch. Psychiat. Nervenkr.*, 183: 163–174, 1949.

[13] G. Pfurtscheller and F. H. L. da Silva, "Event-related EEG/MEG synchronization and desynchronization: basic principles", *Clin. Neurophysiol.*, 110(11): 1842–1857, 1999.

[14] H. Jasper and H. Andrews, "Normal differentiation of occipital and precentral regions in man", *Arch. Neurol. Psychiat. (Chicago)*, 39: 96–115, 1938.

[15] H. Berger, "Über das Elektroenkephalogramm des Menschen", *Arch. Psychiat. Nervenkr.*, 99(6): 555–574, 1933.

[16] F. H. da Silva, T. H. van Lierop, C. F. Schrijer, and W. S. van Leeuwen, "Organization of thalamic and cortical alpha rhythm: Spectra and coherences", *Electroencephalogr. Clin. Neurophysiol.*, 35: 627–640, 1973.

[17] G. Dornhege, B. Blankertz, G. Curio, and K.-R. Müller, "Combining Features for BCI", in: S. Becker, S. Thrun, and K. Obermayer, eds., *Advances in Neural Inf. Proc. Systems (NIPS 02)*, vol. 15, 1115–1122, 2003.

[18] G. Dornhege, B. Blankertz, G. Curio, and K.-R. Müller, "Increase Information Transfer Rates in BCI by CSP Extension to Multi-class", in: S. Thrun, L. Saul, and B. Schölkopf, eds., *Advances in Neural Information Processing Systems*, vol. 16, 733–740, MIT Press, Cambridge, MA, 2004.

[19] K. Fukunaga, *Introduction to Statistical Pattern Recognition*, Academic Press, San Diego, 2nd edn., 1990.

[20] Z. J. Koles and A. C. K. Soong, "EEG source localization: implementing the spatio-temporal decomposition approach", *Electroencephalogr. Clin. Neurophysiol.*, 107: 343–352, 1998.

[21] G. Dornhege, B. Blankertz, G. Curio, and K.-R. Müller, "Boosting bit rates in non-invasive EEG single-trial classifications by feature combination and multi-class paradigms", *IEEE Trans. Biomed. Eng.*, 51(6): 993–1002, 2004.

[22] K.-R. Müller, C. W. Anderson, and G. E. Birch, "Linear and Non-Linear Methods for Brain-Computer Interfaces", *IEEE Trans. Neural Sys. Rehab. Eng.*, 11(2): 165–169, 2003.

[23] B. Blankertz, G. Dornhege, C. Schäfer, R. Krepki, J. Kohlmorgen, K.-R. Müller, V. Kunzmann, F. Losch, and G. Curio, "Boosting Bit Rates and Error Detection for the Classification of Fast-Paced Motor Commands Based on Single-Trial EEG Analysis", *IEEE Trans. Neural Sys. Rehab. Eng.*, 11(2): 127–131, 2003.